# Computer Simulation of Oscillatory Behavior in Cerebral Cortical Networks

Matthew A. Wilson and James M. Bower[1]
Computation and Neural Systems Program
Division of Biology, 216-76
California Institute of Technology
Pasadena, CA 91125

## ABSTRACT

It has been known for many years that specific regions of the working cerebral cortex display periodic variations in correlated cellular activity. While the olfactory system has been the focus of much of this work, similar behavior has recently been observed in primary visual cortex. We have developed models of both the olfactory and visual cortex which replicate the observed oscillatory properties of these networks. Using these models we have examined the dependence of oscillatory behavior on single cell properties and network architectures. We discuss the idea that the oscillatory events recorded from cerebral cortex may be intrinsic to the architecture of cerebral cortex as a whole, and that these rhythmic patterns may be important in coordinating neuronal activity during sensory processing.

## 1    INTRODUCTION

An obvious characteristic of the general behavior of cerebral cortex, as evident in EEG recordings, is its tendency to oscillate. Cortical oscillations have been observed both in the electric fields generated by populations of cells (Bressler and Freeman

1980) as well as in the activity of single cells (Llinas 1988). Our previous efforts to study this behavior involve the construction of realistic, large scale computer simulations of one particular cortical area, the piriform (olfactory) cortex (Wilson and Bower 1989). While the oscillatory behavior of this region has been known for some time (Adrian 1942; Bressler and Freeman 1980), more recent findings of oscillations within visual cortex (Eckhorn et al.,1988; Gray et al. 1989) have generated increased interest in the role of oscillations in cerebral cortex in general. It is particularly intriguing that although these cortical areas receive very different kinds of sensory information, the periodic activity seen in both structures share a common principle frequency component in the range of 30-60 Hz. At the same time, however, the phase relationships of activity across each cortex differ. Piriform cortex displays systematic phase shifts in field potential responses to afferent activation (Freeman 1978; Haberly 1973), while correlations of neuronal activity in visual cortex indicate no such systematic phase shifts (Gray et al. 1989).

In order to compare this oscillatory behavior in these two cortical systems, we have developed a model of visual cortex by modifying the original piriform cortex model to reflect visual cortical network features.

## 2  MODEL STRUCTURE

### 2.1  COMMON MODEL FEATURES

Each simulation has at its base the three basic cell types found throughout cerebral cortex (Figure 1). The principle excitatory neuron, the pyramidal cell, is modeled here as five coupled membrane compartments. In addition there are two inhibitory neurons one principally mediating a slow K+ inhibition and one mediating a fast Cl-inhibition. Both are modeled as a single compartment. Connections between modeled cells are made by axons with finite conduction velocities, but no explicit axonal membrane properties other than delay are included. Synaptic activity is produced by simulating the action-potential triggered release of presynaptic transmitter and the resulting flow of transmembrane current through membrane channels. Each of these channels is described with parameters governing the time course and amplitude of synaptically activated conductance changes. The compartmental models of the cells integrate the transmembrane and axial currents to produce transmembrane voltages. Excursions of the cell body membrane voltage above a specified threshold trigger action potentials. Details of the modeling procedures are described in Wilson and Bower (1989).

Each model is intended to represent a 10 mm x 6 mm cortical region. The many millions of actual neurons in these areas are represented by 375 cells of the three types for a total of 1125 cells. The input to each cortex is provided by 100 independent fibers.

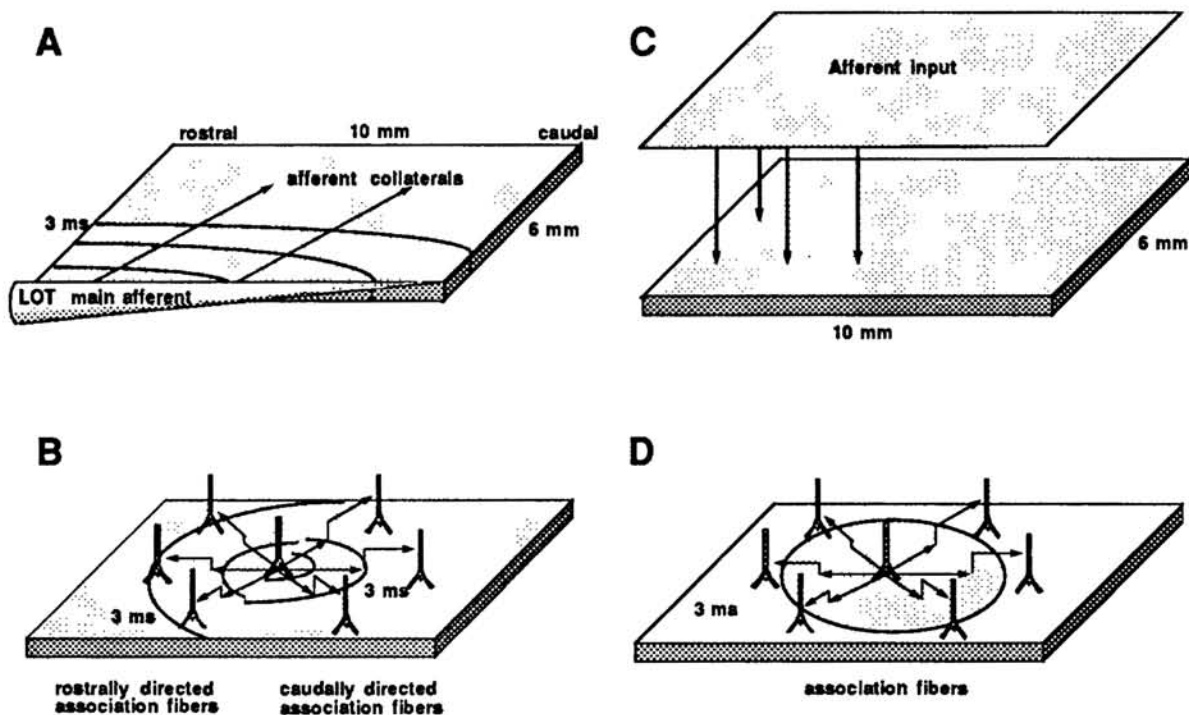

**Figure 1:** In the piriform cortex, input (A) and association fiber (B) projections make distributed lateral contacts with cells over the extent of the cortex. In the visual cortex model, input projections make local contact with cells over a 1 mm radius in a point-to-point fashion (C) and association fibers connect to cells within a limited radius (D).

While both the piriform and visual cortex models reflect these basic features of cerebral cortical architecture, both also contain major structural simplifications. The model referred to as "visual cortex", is particularly simplified. Our objective was to reproduce cortical oscillations characteristic of visual cortex by modifying those basic architectural features that differ between these two brain regions.

## 2.2   MODEL DIFFERENCES

The principle differences between the model of piriform and visual cortex involve changes in the topography of input projections, and in the extent of intrinsic connections within each model. In piriform cortex, afferent input from the olfactory bulb arrives via a tract of axons (LOT) projecting across the surface of the cortex (Fig. 1A) with no topographic relationship between the site of origin of individual LOT axons in the olfactory bulb and their region of termination in the cortex (Haberly 1985). In contrast, projections from the lateral geniculate nucleus to visual cortex are highly topographic, reflecting the retinotopic organization of many structures in the visual system (Van Essen 1979). In piriform cortex, excitatory intrinsic association connections are sparse, distributed, and non-topographic, extending across

the entire cortex (Fig. 1C) (Haberly 1985). In the visual cortex, this association fiber system is much more limited in extent (Gilbert 1983).

## 3   RESULTS

Space limitations do not allow a complete discussion of previous results modeling piriform cortex. Readers are referred to Wilson and Bower (1989) for additional details. Here, we will describe data obtained from the modified piriform cortex model which replicate results from visual cortex.

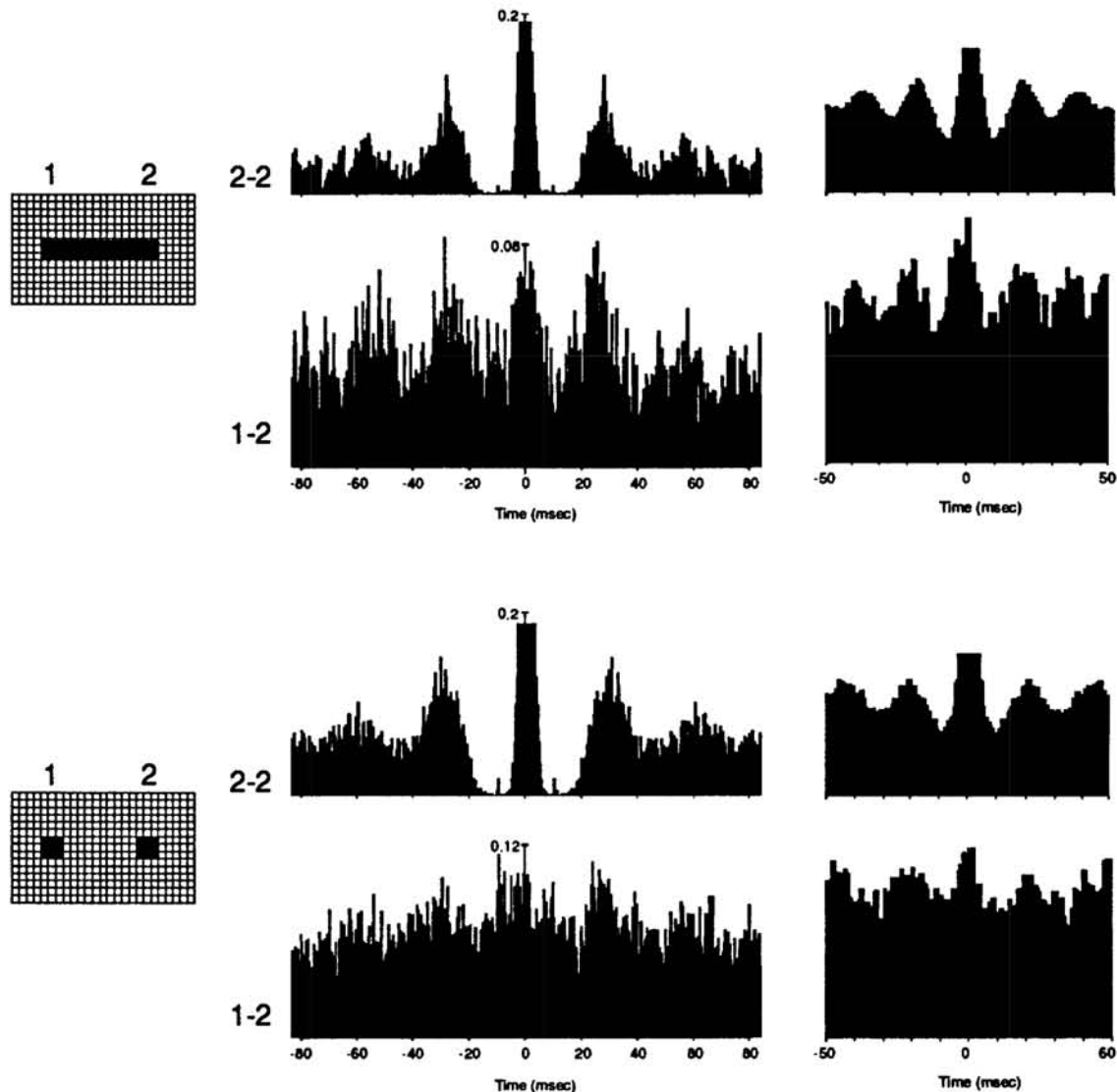

**Figure 2:** Comparison of auto and cross correlations from modeled (middle) and actual (right) (modified from Gray et al. 1989) visual cortex. The left column shows a diagram of the model with the stimulus region shaded. The numbers indicate the location of the recording sites referred to in the auto (2-2) and cross (1-2) correlations. The correlations generated by presentation of a continuous and broken bar stimulus are shown in the upper and lower panels respectively.

Figure 2 shows a comparison of auto and cross correlations of neuronal spike activity taken from both simulated and actual (Gray et al. 1989) experimental data. In each case the two recording sites in visual cortex are separated by approximately 6 mm. Total cross correlations in the modeled data were computed by averaging correlations from 50 individual 500 msec trials. Within each trial simulated activity was generated by providing input representing bars of light at different locations in the visual field. In these cases the model produced oscillatory auto and cross correlations with peak energy in the 30-60 Hz range. As in the experimental data, this effect is most clearly seen when the stimulus is a continuous bar of light activating cells between the two recorded sites (fig. 2). A broken bar which does not stimulate the intermediate region produces a weaker response (fig. 2), again consistent with experimental evidence.

The oscillatory form of the the cross correlation function suggests coherent firing of neurons at the two recorded locations. In order to determine the degree of synchrony between modeled neurons, the difference in phase between the firing of cells in these locations was estimated by measuring the offset of the dominant peak in the cross correlation function. These values were consistent with measurements obtained both through chi-square fitting of a modified sinc function and measurement of the phase of the peak frequency component in the correlation function power spectra. These measurements indicate phase shifts near zero ($< 3$ msec).

## 3.1   STIMULUS EFFECTS

As shown in figure 2, correlations are induced by the presence of a stimulus. However, in both experimental and simulated results these correlations cannot be accounted for through a simple stimulus locking effect. Shuffling the trials with respect to each other prior to calculating cross correlation functions showed oscillations which were greatly diminished or completely absent. At the same time, simulations run in the absence of bar stimuli produced low baseline activity with no oscillations. These results demonstrate that while the stimulus is necessary to induce oscillatory behavior, the coherence between distant points is not due to the stimulus alone.

## 3.2   FREQUENCY

The visual cortex model generates oscillatory neural activity at a frequency in the range of 30-60 Hz, consistent with actual data. As found in the model piriform cortex, the frequency of these oscillations is primarily determined by the time course of the fast feedback inhibitory input. Allowing inhibitory cells to inhibit other inhibitory cells within a local region improved frequency locking and produced auto and cross correlations with more pronounced oscillatory characteristics.

## 3.3   COHERENCE

In order to demonstrate the essential role of the association fiber system in establishing coherent activity, simulations were performed in which all long-range ($> 1$ mm)

association fibers were eliminated. Under these conditions the auto correlations at each recording site continued to show strong oscillatory behavior, but oscillations in the cross correlation function were completely eliminated. Increasing the range of association fibers from 1 to 2 mm restored coherent oscillatory behavior. This demonstrates that long-range association fibers are critical in establishing coherence while local circuitry is sufficient for sustaining oscillations.

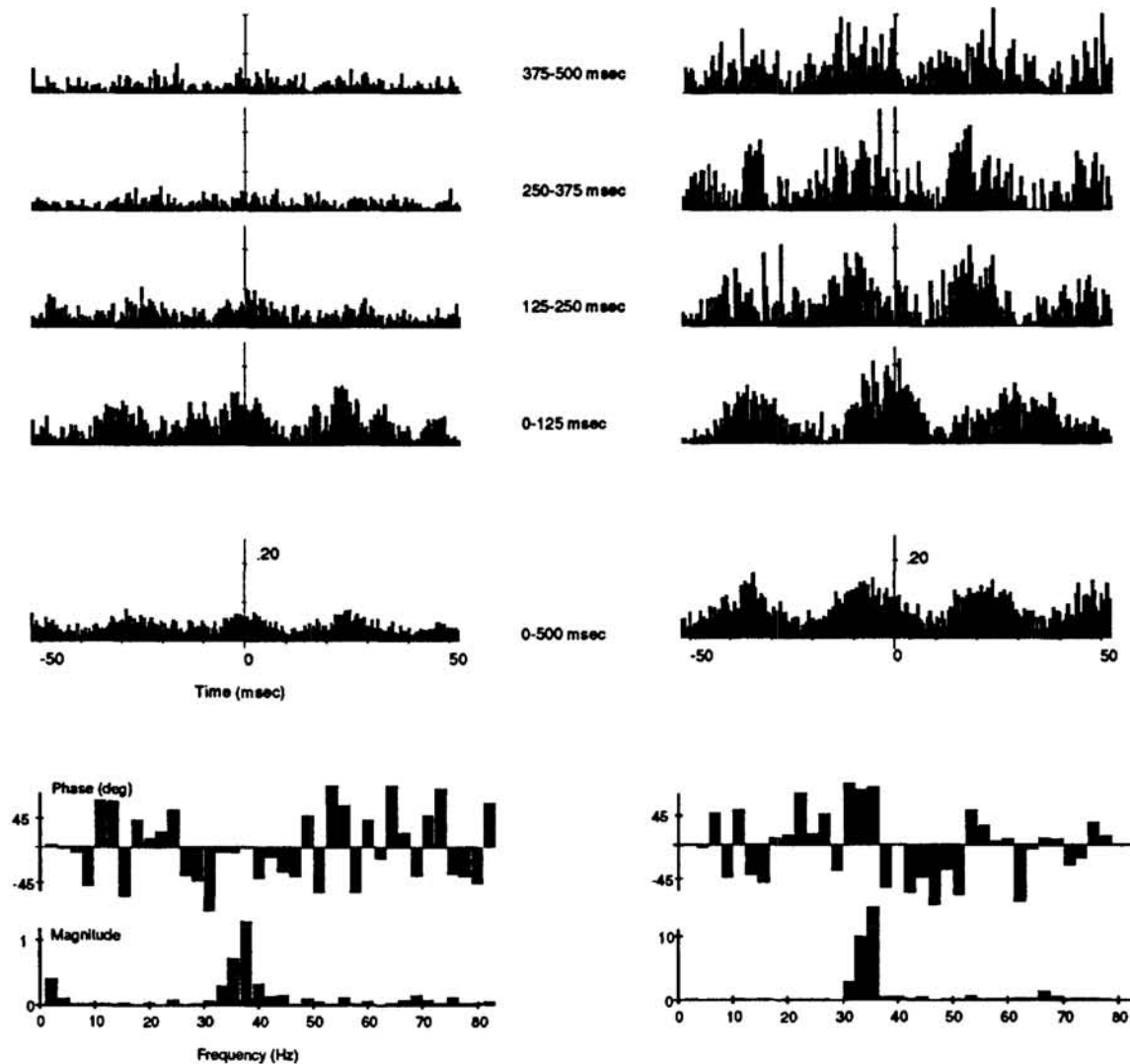

**Figure 3:** Time course of cross correlation functions for relative association fiber coupling strengths of 200 (left) and 300 (right). Upper traces display correlations taken at successive 125 intervals over the 500 msec period. The bottom-most correlation function covers the entire 500 msec interval. The lower panels display the power spectra of the overall correlation function.

### 3.3.1   Association Fiber Delay

To examine the dependence of zero-phase coherence between distant sites on association fibers characteristics, the propagation velocity for spikes travelling between pyramidal cells was reduced from a mean of 0.86 m/s to 0.43 m/s. Under these conditions the phase shift in the cross correlation function for a continuous bar stimulus remained less than 3 msec. This result indicates that the zero-phase coherence is not a direct function of association fiber delays.

### 3.3.2   Coupling Strength

As shown in figure 3, increasing the degree of association fiber coupling by increasing synaptic weights produced a transition from zero-phase coherence to a coherence with an 8 msec phase shift. Intermediate shifts were not observed. Figure 3 also illustrates the time course of coherence and phase relationships. There is a tendency for the initial stimulus onset period (0-125 msec) to show zero-phase preference. Later periods (> 125 msec) reflect the association coupling induced phase shift. For weak coupling which produces zero-phase behavior, the correlation structure decays over the 500 msec stimulus period. Increased coupling strength provides more sustained coherence, as does the addition of mutual inhibition.

## 4   DISCUSSION

Analysis of the behavior of the models shows that several components are particularly important in establishing the different phase and frequency relationships. A key factor in establishing zero-phase coherence appears to be the stimulation of a cellular population which can activate, via association fibers, adjacent regions in a symmetric fashion. In the case of the continuous bar, this intermediate region lies in the center of the bar. This is consistent with experimental results which indicate reduced coherence with bar stimuli which do not excite this region. The model also indicates that frequency can be effectively modulated by inhibitory feedback. The fact that inhibitory events with similar temporal properties are found throughout the cerebral cortex suggests that oscillations in the 30-60 Hz range will be found in a number of different cortical areas.

Interpreting phase coherence from correlation functions produced from the average of many simulation trials pointed out the need to distinguish average phase effects from instantaneous phase effects. Instantaneous phase implies that the statistics of the correlation function taken at any point of any trial are consistent with the statistics of the combined data. Average phase allows for systematic within-trial and between-trial variability and is, therefore, a weaker assertion of actual coherence. This distinction is particularly important for theories which rely on phase encoding of stimulus information. Analysis of our model results indicates that the observed phase relationships are an average rather than an instantaneous effect.

Based on previous observations of the behavior of the piriform cortex model, we have proposed that high frequency oscillations may reflect the gating of intrinsic

network integration intervals. This modulatory role would serve to assure that cells do not fire before they have received the necessary input to initiate another round of cortical activity. While this is clearly only one possible functional role for oscillations in piriform cortex, the model is being used to extend this idea to processing in the visual cortex as well.

## Acknowledgements

This research was supported by the NSF (EET-8700064), the ONR (Contract N00014-88-K-0513), and the Lockheed Corporation.

## Footnotes

[1] Please address correspondence to James M. Bower at above address.

## References

Adrian, E.D. 1942. Olfactory reactions in the brain of the hedgehog. J. Physiol. (Lond.) 100, 459-472.

Bressler, S.L. and W.J. Freeman. 1980. Frequency analysis of olfactory system EEG in cat, rabbit and rat. Electroenceph. clin. Neurophysiol. 50, 19-24.

Eckhorn, R., R. Bauer, Jordan, M. Brosch, W. Kruse, M. Munk, and H.J. Reitboeck. 1988. Coherent oscillations: A mechanism of feature linking in the visual cortex? Biol. Cybern. 60, 121-130.

Freeman, W.J. 1978. Spatial properties of an EEG event in the olfactory bulb and cortex. Electroenceph. clin. Neurophysiol. 44,586-605.

Gilbert, C.D. 1983. Microcircuitry of the visual cortex. Ann. Rev. Neurosci. 6,217-247.

Gray, C.M., P. Konig, A.K. Engel, W. Singer. 1989. Oscillatory responses in cat visual cortex exhibit inter-columnar synchronization which reflects global stimulus properties. Nature 338, 334-337.

Haberly, L.B. 1985. Neuronal circuitry in olfactory cortex: anatomy and functional implications. Chem. Senses 10, 219-238.

Haberly, L.B. 1973. Summed potentials evoked in opossum prepyriform cortex. J. Neurophysiol. 36, 775-788.

Kammen, D.M., P.J. Holmes, and C. Koch. 1989. Cortical architecture and oscillations in neuronal networks: Feedback versus local coupling. In: Models of Brain Function R.M.J. Cotterill, Ed. (Cambridge Univ. Press.)

Llinas, R. 1988. The intrinsic electrophysiological properties of mammalian neurons: Insights into central nervous system function. Science 242:1654-1664.

Wilson, M.A. and J.M Bower. 1989. The simulation of large scale neuronal networks. In Methods in Neuronal Modeling: From Synapses to Networks C. Koch and I. Segev, Eds. (MIT Press, Cambridge, MA.) pp. 291-334.

Van Essen, D.C. 1979. Visual areas of the mammalian cerebral cortex. Ann. Rev. Neurosci. 2, 227-263.